# Bayesian Inference for Spiking Neuron Models with a Sparsity Prior

**Sebastian Gerwinn**         **Jakob H Macke**         **Matthias Seeger**

**Matthias Bethge**
Max Planck Institute for Biological Cybernetics
Spemannstrasse 41
72076 Tuebingen, Germany
`{firstname.surname}@tuebingen.mpg.de`

## Abstract

Generalized linear models are the most commonly used tools to describe the stimulus selectivity of sensory neurons. Here we present a Bayesian treatment of such models. Using the expectation propagation algorithm, we are able to approximate the full posterior distribution over all weights. In addition, we use a Laplacian prior to favor sparse solutions. Therefore, stimulus features that do not critically influence neural activity will be assigned zero weights and thus be effectively excluded by the model. This feature selection mechanism facilitates both the interpretation of the neuron model as well as its predictive abilities. The posterior distribution can be used to obtain confidence intervals which makes it possible to assess the statistical significance of the solution. In neural data analysis, the available amount of experimental measurements is often limited whereas the parameter space is large. In such a situation, both regularization by a sparsity prior and uncertainty estimates for the model parameters are essential. We apply our method to multi-electrode recordings of retinal ganglion cells and use our uncertainty estimate to test the statistical significance of functional couplings between neurons. Furthermore we used the sparsity of the Laplace prior to select those filters from a spike-triggered covariance analysis that are most informative about the neural response.

## 1   Introduction

A central goal of systems neuroscience is to identify the functional relationship between environmental stimuli and a neural response. Given an arbitrary stimulus we would like to predict the neural response as well as possible. In order to achieve this goal with limited amount of data, it is essential to combine the information in the data with prior knowledge about neural function. To this end, generalized linear models (GLMs) have proven to be particularly useful as they allow for flexible model architectures while still being tractable for estimation.

The GLM neuron model consists of a linear filter, a static nonlinear transfer function and a Poisson spike generating mechanism. To determine the neural response to a given stimulus, the stimulus is first convolved with the linear filter (i.e. the receptive field of the neuron). Subsequently, the filter output is converted into an instantaneous firing rate via a static nonlinear transfer function, and finally spikes are generated from an inhomogeneous Poisson-process according to this firing rate. Note, however, that the GLM neuron model is not limited to describe neurons with Poisson firing statistics. Rather, it is possible to incorporate influences of its own spiking history on the neural response. That is, the firing rate is then determined by a combination of both the external

stimulus and the spiking-history of the neuron. Thus, the model can account for typical effects such as refractory periods, bursting behavior or spike-frequency adaptation. Last but not least, the GLM neuron model can also be applied for populations of coupled neurons by making each neuron dependent not only on its own spiking activity but also on the spiking history of all the other neurons.

In previous work (Pillow et al., 2005; Chornoboy et al., 1988; Okatan et al., 2005) it has been shown how point-estimates of the GLM-parameters can be obtained using maximum-likelihood (or maximum a posteriori (MAP)) techniques. Here, we extend this approach one step further by using Bayesian inference methods in order to obtain an approximation to the full posterior distribution, rather than point estimates. In particular, the posterior determines confidence intervals for every linear weight, which facilitates the interpretation of the model and its parameters. For example, if a weight describes the strength of coupling between two neurons, then we can use these confidence intervals to test whether this weight is significantly different from zero. In this way, we can readily distinguish statistical significant interactions between neurons from spurious couplings.

Another application of the Bayesian GLM neuron model arises in the context of spike-triggered covariance analysis. Spike-triggered covariance basically employs a quadratic expansion of the external stimulus parameter space and is often used in order to determine the most informative subspace. By combining spike-triggered covariance analysis with the Bayesian GLM framework, we will present a new method for selecting the filters of this subspace.

Feature selection in the GLM neuron model can be done by the assumption of a Laplace prior over the linear weights, which naturally leads to sparse posterior solutions. Consequently, all weights are equally strongly pushed to zero. This contrasts the Gaussian prior which pushes weights to zero proportional to their absolute value. In this sense, the Laplace prior can also be seen as an efficient regularizer, which is well suited for the situation when a large range of alternative explanations for the neural response shall be compared on the basis of limited data. As we do not perform gradient descent on the posterior, differentiability of the posterior is not required.

The paper is organized as follows: In section 2, we describe the model, and the "expectation propagation" algorithm (Minka, 2001; Opper & Winther, 2000) used to find the approximate posterior distribution. In section 3, we estimate the receptive fields, spike-history effects and functional couplings of a small population of retinal ganglion cells. We demonstrate that for small training sets, the Laplace-prior leads to superior performance compared to a Gaussian-prior, which does not lead to sparse solutions. We use the confidence intervals to test whether the functional couplings between the neurons are significant.

In section 4, we use the GLM neuron model to describe a complex cell response recorded in macaque primary visual cortex: After computing the spike-triggered covariance (STC) we determine the relevant stimulus subspace via feature selection in our model. In contrast to the usual approach, the selection of the subspace in our case becomes directly linked to an explicit neuron model which also takes into account the spike-history dependence of the spike generation.

## 2 Generalized Linear Models and Expectation Propagation

### 2.1 Generalized Linear Models

Let $X_t \in \mathbb{R}^d, t \in [0, T]$ denote a time-varying stimulus and $D_i = \{t_{i,j}\}$ the spike-times of $i = 1, \ldots, n$ neurons. Here $X_t$ consists of the sensory input at time $t$ and can include preceeding input frames as well. We assume that the stimulus can only change at distinct time points, but can be evaluated at continous time $t$. We would like to incorporate spike-history effects, couplings between neurons and dependence on nonlinear features of the stimulus. Therefore, we describe the effective input to a neuron via the following feature-map:

$$\psi(t) = \psi_{st}(X_t) \bigoplus_i \psi_{sp}(\{t_{i,j} \in D_i : t_{i,j} < t\}),$$

where $\psi_{sp}$ represents the spike time history and $\psi_{st}$ the possibly nonlinear feature map for the stimulus. That is, the complete feature vector $\psi$ contains possibly nonlinear features of the stimulus and the spike history of every neuron. Any feature which is causal in the sense that it does not depend on future events can be used. We model the spike history dependence by a set of small time

windows $[t - \tau_l, t - \tau_l')$ in which occuring spikes are counted.

$$(\psi_{sp,i}(\{t_{i,j} \in D_i : t_{i,j} < t\}))_l \quad = \quad \sum_{j:t_{i,j}<t} \mathbb{1}_{[t-\tau_l,t-\tau_l')}(t_{i,j}) \quad ,$$

where $\mathbb{1}_{[a,b)}(t)$ denotes the indicator function which is one if $t \in [a,b)$ and zero otherwise. In other words, for each neuron there is a set of windows $l = 1, \ldots, L$ with time-lags $\tau_l$ and width $\tau_l - \tau_l'$ describing its spiking history. More precisely, the rate can only change if the stimulus changes or a spike leaves or enters one of these windows. Thus, we obtain a sequence of *changepoints* $0 = \tilde{t}_0 < \tilde{t}_1 < \cdots < \tilde{t}_j < \cdots < T$, where each feature $\psi_i(t)$ is constant in $[\tilde{t}_{j-1}, \tilde{t}_j)$, attaining the value $\psi_{i,j}$. In the GLM neuron model setting the instantanious firing rate of neuron $i$ is obtained by a linear filter of the feature map:

$$p(\text{spike}|X_t, \{t_{i,j} \in D : t_{i,j} < t\}) \quad = \quad \lambda(\mathbf{w}_i^T \psi(t)), \tag{1}$$

where $\lambda$ is the nonlinear transfer function. Following general point process theory (Snyder & Miller, 1991) and using the fact that the features stay constant between two changepoints we can write down the likelihood $P(D|\{\mathbf{w}\}) = \prod_{i=1}^n L_i(\mathbf{w}_i)$, where each $L_i(\mathbf{w}_i)$ has the form

$$L_i(\mathbf{w}_i) \quad \propto \quad \prod_j \phi_{i,j}(u_{i,j}), \quad u_{i,j} = \mathbf{w}_i^T \psi_j,$$

$$\phi_{i,j}(u_{i,j}) \quad = \quad \lambda_i(u_{i,j})^{\sum_{t \in D_i} \delta(t-\tilde{t}_j)} \exp(-\lambda_i(u_{i,j})(\tilde{t}_j - \tilde{t}_{j-1})) \quad .$$

The function $\delta(.)$ in the second equation is defined to be one if and only if its argument equals zero. The sum therfore is 1 iff a spike of neuron $i$ occurs at changepoint $\tilde{t}_j$. Note that the changepoints $\tilde{t}_j$ depend on the spikes and therefore, the process is not Poissonian, as it might be suggested by the functional form of the likelihood.

As it has been shown in (Paninski, 2004), the likelihood is log-concave in $\mathbf{w}_i$ if $\lambda_i(\cdot)$ is both convex and log-concave. We are using the transfer function $\lambda_i(u) = e^u$ which, in particular, gives rise to a log-linear point process model. Alternatively, one could also use $\lambda_i(u) = e^u \mathbb{1}_{u<0} + (1+u)\mathbb{1}_{u\geq 0}$, which grows only linearly (cf. Harris et al. (2003); Pillow et al. (2005)).

While we require all rates $\lambda_i(t)$ to be piecewise constant, it should be noted that we do not restrict ourselves to a uniform quantization of the time axis. In this way, we achieve an efficient architecture for which the density of change points automatically adapts to the speed with which the input signal is changing.

The choice of the prior distribution can play a central role when coping with limited amount of data. We use a Laplace prior distribution over the weights in order to favor sparse solutions over those which explain the data equally well but require more weights different from zero (c.f. Tibshirani (1996)):

$$P(\mathbf{w}_i) \propto \prod_k e^{-\rho_k|w_{k,i}|}. \tag{2}$$

Thus, prior factors have the form $\phi_{i,k}(u_{i,k}) = \frac{\rho_k}{2} \exp(-\rho_k|u_{i,k}|)$ with $\psi_k = (\mathbb{1}_{l=k})_l$ and $u_{i,k} = \mathbf{w}_i^T \psi_k$ as above. In our applications, we allowed the prior variance $\frac{2}{\rho_k^2}$ of the stimulus-dependent features to be different from the variance of the spike-history features. The posterior takes the form:

$$P(\mathbf{w}|D) \quad \propto \quad \prod_{i,j} \phi_{i,j}(u_{i,j}),$$

where each $\phi_{i,j}$ individually instantiates a Generalized Linear Model (either corresponding to a likelihood factor or to a prior factor). As the posterior factorizes over neurons, we can perform our analysis for each neuron seperately. Therefore, for simplicity we drop the subscript $i$ in the following.

Our model does not assume or require any specific stimulus distribution. In particular, it is not limited to white noise stimuli or elliptically contoured distributions but it can be used without modification for other stimulus distributions such as natural image sequences. Finally, this framework allows exact sampling of spike trains due to the piecewise constant rate.

## 2.2 Expectation Propagation

As exact Bayesian inference is intractable in our model, we seek to find a good approximation to the full posterior. In our case all likelihood and prior factors are log-concave. Therefore, the posterior is unimodal and a Gaussian approximation is well suited. A frequently used technique for this purpose is the Laplace-approximation which computes a quadratic approximation to the log-posterior based on the Hessian around the maximum. For the Laplacian prior, however, this approach falls short since the distribution is not differentiable at zero. Instead, we employ the Expectation Propagation (EP) algorithm (Minka, 2001; Opper & Winther, 2000). In this approximation technique, each factor (also called *site*) $\phi_j$ of the posterior is replaced by an unnormalised Gaussian:

$$N^U(u_j|b_j, \pi_j) = \exp(-\frac{1}{2}\pi_j u_j^2 + b_j u_j) =: \hat{\phi}(u_j), \quad \pi_j \geq 0$$

where the $b_j$, $\pi_j$ are called the *site parameters*. The approximation aims at minimizing the Kullback-Leibler divergence between the full posterior $P(\mathbf{w}|D)$ and the approximation, $Q(\mathbf{w}) \approx \prod_j \hat{\phi}(u_j)$. The log-concavity of the model implies that all $\pi_j \geq 0$, which supports the numerical stability of the EP algorithm. Some of the $\pi_j$ may even be 0, as long as $Q(\mathbf{w})$ is a (normalizable) Gaussian. An EP update at $j$ consists of computing the Gaussian *cavity distribution* $Q^{\setminus j} \propto Q\hat{\phi}_j^{-1}$ and the non-Gaussian *tilted distribution* $\hat{P} \propto Q^{\setminus j}\phi_j$, then updating $b_j$, $\pi_j$ such that the new $Q'$ has the same mean and covariance as $\hat{P}$ (moment matching). This is iterated in random order over the sites until convergence.

We omit the detailed update schemes here and refer to (Seeger et al., 2007; Seeger, 2005). Convergence guarantees for EP applied to non-Gaussian log-concave models have not been shown so far. Nevertheless it is reported that at least in the log-concave case EP behaves stable (e.g., Rasmussen & Williams (2006)), and we observe quick convergence in our case ( $\leq 20$ iterations over all sites are required). The model still contains hyperparameters, namely the prior variances $\frac{2}{\rho_k^2}$. In each experiment, these were determined via a standard crossvalidation procedure (80% training data, 10% validation, 10% test).

## 3 Modeling retinal ganglion cells: Which cells are functionally coupled?

We applied the GLM neuron model to multi-electrode recordings of three rabbit retinal ganglion cells. The stimulus consisted of 32767 frames each of which showing a random $16 \times 16$ checkerboard pattern with a refresh rate of 50 Hz (data provided by G. Zeck, see (Zeck et al., 2005)).

First, in order to investigate the role of the Laplace prior, we trained a single cell GLM neuron model on datasets of different sizes with either a Laplace prior or a Gaussian prior. The models, which have the same number of parameters, were compared by evaluating their negative log-likelihood on an independent test set. As can be seen on the right the choice of prior becomes less important for large training sets as the weights are sufficiently constrained by the data. For each training set size a separate crossvalidation was carried out. Errorbars were obtained by drawing 100 samples from the posterior.

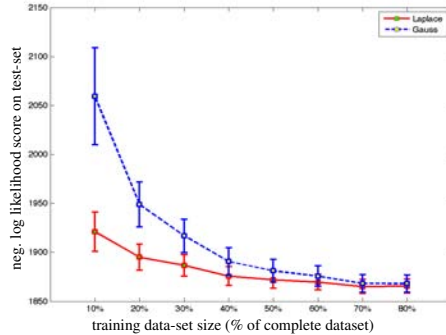

Fig. 1 shows the spatiotemporal receptive field of each neuron, as well as the filters describing the influence of spiking history and input from other cells. For conciseness, we only plot the filters for 80 and 120 ms time lags, but the fitted model included 60 and 140 ms time lags as well. The strongly positive weights on the diagonal of figure 1(c) for the spiking history can be interpreted as "self-excitation". In this way, it is possible to model the bursting behavior exhibited by the cells in our recordings (see also Fig. 2). The strongly negative weights at small time lags represent refractory periods. The red lines correspond to 3 standard deviations of the posterior. The first neuron seems to elicit "bursts" at lower frequencies. Note the different scaling of the y-axis for diagonal and off-diagonal terms. By analyzing the coupling terms, we can see that there is significant interaction

between cells 2 and 3, but not between any other pair of cells. As our prior assumption is that the couplings are 0, this interaction-term is not merely a consequence of our choice of prior. As a result of our crossvalidation it turns out that the prior variance for spike history weights should be set to very large values ($\rho$= 0.1, variance = $2\frac{1}{\rho^2}$) meaning that these are well determined by the data. In contrast, prior variances for the stimulus weights should be more strongly biased towards zero ($\rho$ = 150).

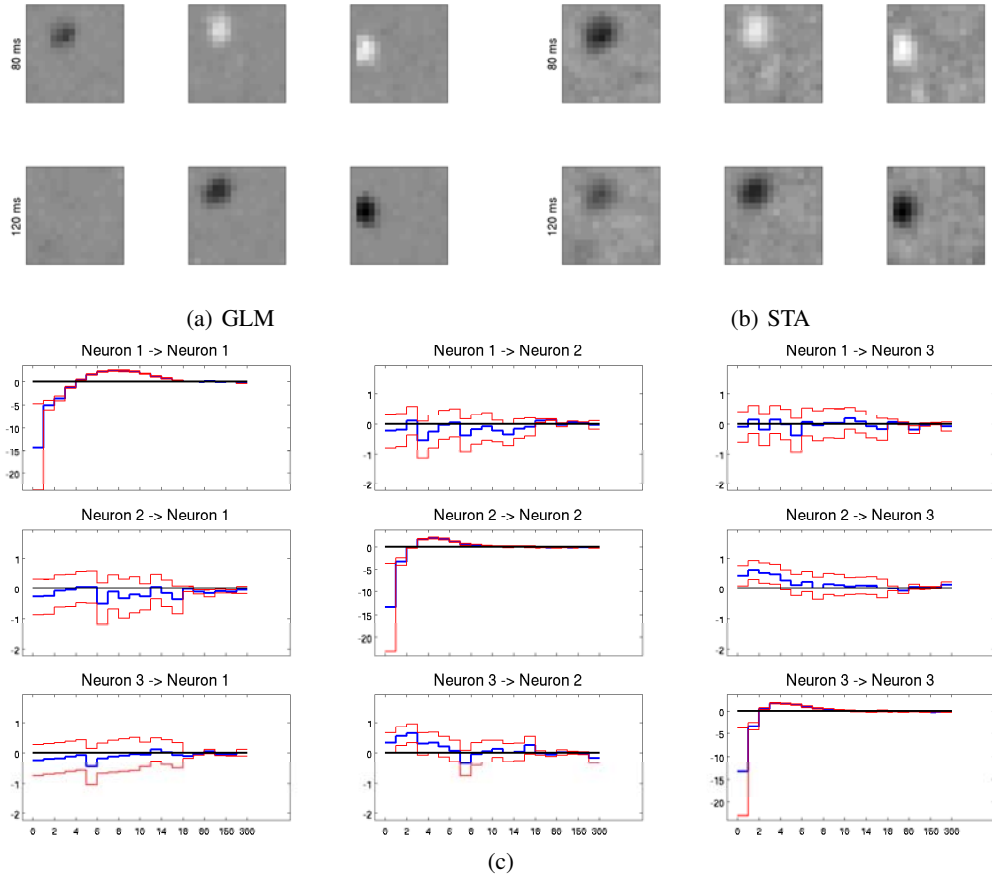

Figure 1: (a): Stimulus dependence inferred by the GLM for the three neurons (columns) at different time lags (rows). 2 of 4 time lags are plotted (60, 140 ms not shown). (b): Spike-triggered average for the same neurons and time lags as in (a). (c): Causal dependencies between the three neurons. Each plot shows the value of the linear weight as a function of increasing time lag $\tau_l$ (in ms). Shown are posterior mean and three std. dev. (indicated in red). Different scaling of the y-axis is used for diagonal and off-diagonal plots.

|  | STA | GLM | GLM with couplings |
|---|---|---|---|
| Neuron 1 | 0.2199 | 0.2442 | 0.3576 |
| Neuron 2 | 0.1746 | 0.2348 | 0.3320 |
| Neuron 3 | 0.1828 | 0.3319 | 0.4202 |
| Mean | 0.1924 | 0.2703 | 0.3699 |

Table 1: Predictions performance of different models. Entries correspond to the correlation coefficient between the predicted rate of each model and spikes on a test set. Both rate and spikes are binned in 5 ms bins. The first GLM models neither connections nor self-feedback.

Because of the regularization by the prior the spatio-temporal receptive fields are much smoother than the spike-triggered average ones, see Fig. 1(a). The receptive fields of the STA seems to be

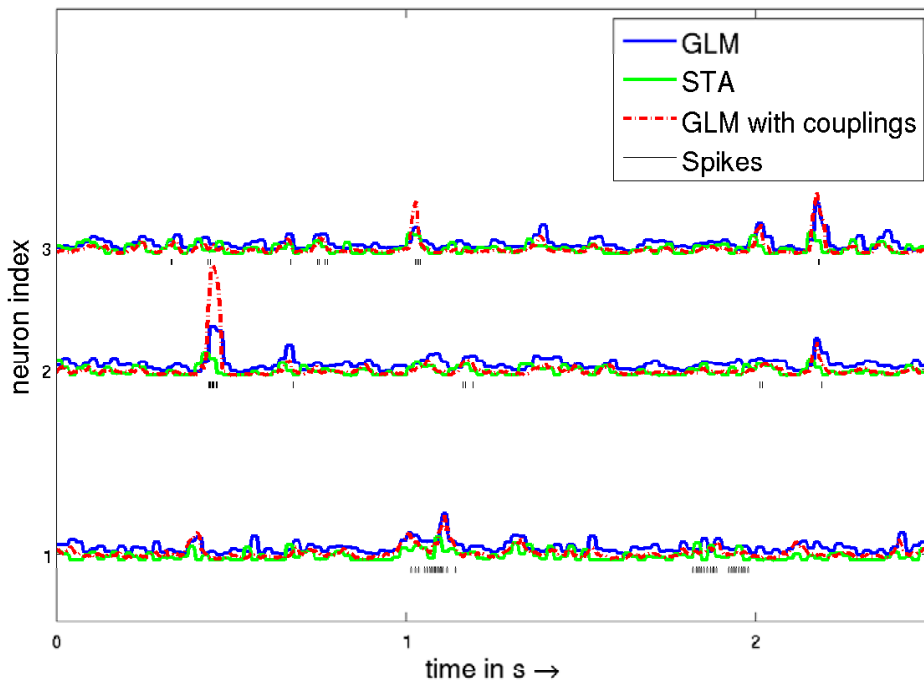

Figure 2: Predicted rate for the GLM neuron model with and without any spike history and the predicted rate for the STA for the same neurons as in the other plots. For the STA the linear response is rectified. Rate for the GLM with spike dependence is obtained by averaging over 1000 sampled spike-trains. Rates are rescaled to have the same standard deviation.

more smeared out which might be due to the fact that it cannot model bursting behavior. The more conservative estimate of the sparse neuron model should increase the prediction performance. To verify this, we calculated the linear response from the spike-triggered average and the rate of our GLM neuron model. In order to have the same number of parameters we neglected all connections. As a model free performance measure we used the correlation coefficient between the spike trains and the rates (each are binned in 5 ms bins). For the GLM with couplings, rates were estimated by sampling 1000 spike trains with the posterior mean as linear weights. As our model explicitly includes the nonlinearity during fitting, the rate is more sharply peaked around the spikes, see Fig. 2. The prediction performance can be increased even further by modeling couplings between neurons as summarized in Tab. 1.

## 4 Modeling complex cells: How many filters do we need?

Complex cells in primary visual cortex exhibit strongly nonlinear response properties which cannot be well described by a single linear filter, but rather requires a set of filters. A common approach for finding these filters is based on the covariance of the spike-triggered ensemble: Eigenvectors of eigenvalues that are much bigger (or smaller) than the eigenvalues of the whole stimulus ensemble indicate directions in stimulus space to which the cell is sensitive to. Usually, a statistical hypothesis test on the eigenvalue-spectrum is used to decide how many of the eigenvectors $e_i$ are needed to model the cells (Simoncelli et al., 2004; Touryan et al., 2002; Rust et al., 2005; Steveninck & Bialek, 1988). Here, we take a different approach: We use the confidence intervals of our GLM neuron model to determine the relevant dimensions within the subspace revealed by STC. We first apply STC to find the space spanned by a set of eigenvectors that is substantially larger than the expected dimensionality of the relevant subspace. Next, we fit a nonlinear function $n_i$ to the filter-outputs $f_i(X_t) = \langle X_t, e_i \rangle$. Finally, we linearly combine the $n_i(t)$, resulting in a model of the same form as equation (1) with $(\psi_{st})_i(X_t) = n_i(f_i(X_t))$

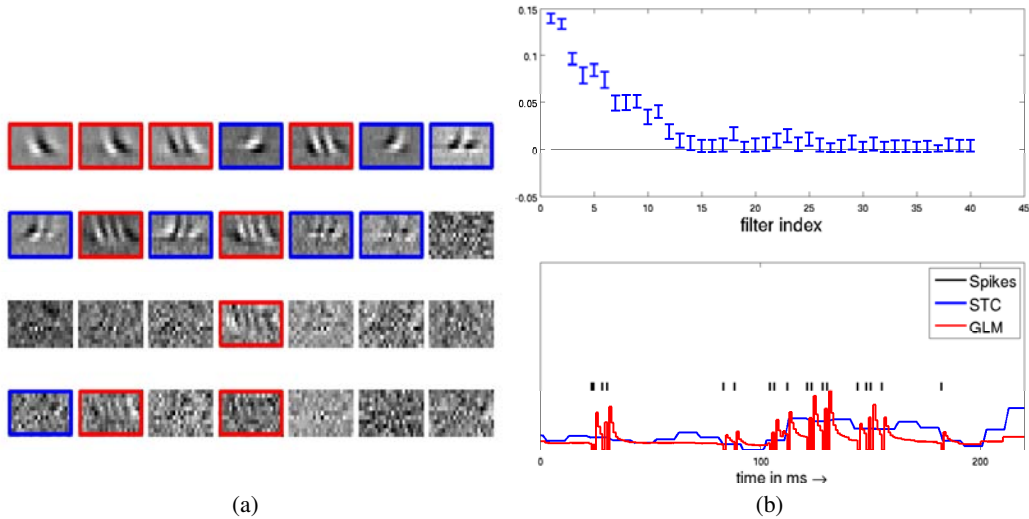

(a)                                              (b)

Figure 3: (a): 24 out of 40 Filters estimated by STC. The filters are ordered according to their log-ratio of their eigenvalue to the corresponding eigenvalue of the complete stimulus ensemble (from left to right). Highlighted filter are those with significant non-zero weights, red indicating excitatory and blue inhibitory filters. (b) Upper: Posterior mean +/- 3 std. dev. Filter indices are ordered in the same way as in (a). Lower: Predicted rate on a test set for STC and for the GLM neuron model with spike history dependence on a test set.

As the model is linear in the weights $w_i$, we can use the GLM neuron model to fit these weights and obtain confidence intervals. If a filter $f_i(t)$ is not needed for explaining the cells response, its corresponding weight $w_i$ will automatically be set to zero by the model due to the sparsity prior. This provides an alternative, model-based method of determining the number of filters required to model the cell. The significance of each filter is not determined by a separate hypothesis test on the spectrum of the spike-triggered covariance, but rather by assessing its influence on the neural activity within the full model.

As in the previous application, we can model the spike history effects with an additional feature vector $\psi_{sp}$ to take into account temporal dynamics of single neurons or couplings.

Before applying our method to real data, we tested it on data generated from an artificial complex cell similar to the one in (Rust et al., 2005). On this simulated data we were able to recover the original filters. We then fitted this GLM neuron model to data recorded from a complex cell in primary visual cortex of an anesthetized macaque monkey (same data as in (Rust et al., 2005)). We first extracted 40 filters which eigenvalues were most different to their corresponding eigenvalues of the complete stimulus ensemble. Any nonlinear regression procedure could be used to fit a nonlinearity to each filter output. We used a simple quadratic regression technique. Having fixed the first nonlinearity we approximated the posterior as above. The resulting confidence intervals for the linear weights are plotted in Fig. 3(b). The filters with significant non-zero weights are highlighted in Fig. 3(a). Red indicates exitatory and blue inhibitory effects on the firing rate. Using 3 std. dev. confidence intervals 9 excitatory and 8 inhibitory filters turned out to be significant in our model. The number of filters is similar to that reportet in Rust et al., who regarded 7 excitatory and 7 inhibitory filters as significant (Rust et al., 2005). The rank order of the linear weights is closely related but not identical to the order of eigenvalues, as can be seen in Fig. 3(b), top.

## 5   Summary and Conclusions

We have shown how approximate Bayesian inference within the framework of generalized linear models can be used to address the problem of identifying relevant features of neural data. More precisely, the use of a sparsity prior favors sparse posterior solutions: non-zero weights are assigned only to those features which which are critical for explaining the data. Furthermore, the explicit

uncertainty information obtained from the posterior distribution enables us to identify ranges of statistical significance and therefore facilitates the interpretation of the solution. We used this technique to determine couplings between neurons in a multi-cell recording and demonstrated an increase in prediction performance due to regularization by the sparsity prior. Also, in the context of spike-triggered covariance analysis, we used our method to determine the relevant stimulus subspace within the space spanned by the eigenvectors. Our subspace selection method is directly linked to an explicit neuron model which also takes into account the spike-history dependence of the spike generation.

### Acknowledgements

We would like to thank Günther Zeck and Nicole Rust for generously providing their data and for useful discussions.

# References

Chornoboy, E., Schramm, L., & Karr, A. (1988). Maximum likelihood identification of neural point process systems. *Biological Cybernetics*, *59*, 265-275.

Harris, K., Csicsvari, J., Hirase, H., Dragoi, G., & Buzsaki, G. (2003). Organization of cell assemblies in the hippocampus. *Nature*, *424*(6948), 552–6.

Minka, T. (2001). Expectation propagation for approximate Bayesian inference. *Uncertainty in Artificial Intelligence*, *17*, 362–369.

Okatan, M., Wilson, M. A., & Brown, E. N. (2005). Analyzing functional connectivity using a network likelihood model of ensemble neural spiking activity. *Neural Computation*, *17*, 1927-1961.

Opper, M., & Winther, O. (2000). Gaussian Processes for Classification: Mean-Field Algorithms. *Neural Computation*, *12*(11), 2655-2684.

Paninski, L. (2004). Maximum likelihood estimation of cascade point-process neural encoding models. *Network*, *15*(4), 243–262.

Pillow, J. W., Paninski, L., Uzzell, V. J., Simoncelli, E. P., & Chichilnisky, E. J. (2005). Prediction and decoding of retinal ganglion cell responses with a probabilistic spiking model. *J Neurosci*, *25*(47), 11003–11013.

Rasmussen, C., & Williams, C. (2006). *Gaussian processes for machine learning*. Springer.

Rust, N., Schwartz, O., Movshon, J., & Simoncelli, E. (2005). Spatiotemporal Elements of Macaque V1 Receptive Fields. *Neuron*, *46*(6), 945–956.

Seeger, M. (2005). *Expectation propagation for exponential families* (Tech. Rep.). University of California at Berkeley. (See www.kyb.tuebingen.mpg.de/bs/people/seeger.)

Seeger, M., Steinke, F., & Tsuda, K. (2007). Bayesian inference and optimal design in the sparse linear model. *AI and Statistics*.

Simoncelli, E., Paninski, L., Pillow, J., & Schwartz, O. (2004). Characterization of neural responses with stochastic stimuli. In M. Gazzaniga (Ed.), (Vol. 3, pp. 327–338). MIT Press.

Snyder, D., & Miller, M. (1991). *Random point processes in time and space.* Springer Texts in Electrical Engineering.

Steveninck, R., & Bialek, W. (1988). Real-Time Performance of a Movement-Sensitive Neuron in the Blowfly Visual System: Coding and Information Transfer in Short Spike Sequences. *Proceedings of the Royal Society of London. Series B, Biological Sciences*, *234*(1277), 379–414.

Tibshirani, R. (1996). Regression Shrinkage and Selection via the Lasso. *Journal of the Royal Statistical Society. Series B (Methodological)*, *58*(1), 267–288.

Touryan, J., Lau, B., & Dan, Y. (2002). Isolation of Relevant Visual Features from Random Stimuli for Cortical Complex Cells. *Journal of Neuroscience*, *22*(24), 10811.

Zeck, G. M., Xiao, Q., & Masland, R. H. (2005). The spatial filtering properties of local edge detectors and brisk-sustained retinal ganglion cells. *Eur J Neurosci*, *22*(8), 2016-26.

